# An Analog VLSI Saccadic Eye Movement System

Timothy K. Horiuchi     Brooks Bishofberger     and     Christof Koch
Computation and Neural Systems Program
California Institute of Technology
MS 139-74
Pasadena, CA 91125

## Abstract

In an effort to understand saccadic eye movements and their relation to visual attention and other forms of eye movements, we — in collaboration with a number of other laboratories — are carrying out a large-scale effort to design and build a complete primate oculomotor system using analog CMOS VLSI technology. Using this technology, a low power, compact, multi-chip system has been built which works in real-time using real-world visual inputs. We describe in this paper the performance of an early version of such a system including a 1-D array of photoreceptors mimicking the retina, a circuit computing the mean location of activity representing the superior colliculus, a saccadic burst generator, and a one degree-of-freedom rotational platform which models the dynamic properties of the primate oculomotor plant.

## 1   Introduction

When we look around our environment, we move our eyes to center and stabilize objects of interest onto our fovea. In order to achieve this, our eyes move in quick jumps with short pauses in between. These quick jumps (up to 750 *deg/sec* in humans) are known as **saccades** and are seen in both exploratory eye movements and as reflexive eye movements in response to sudden visual, auditory, or somatosensory stimuli. Since the intent of the saccade is to bring new objects of interest onto the fovea, it can be considered a primitive attentional mechanism. Our interest

lies in understanding how saccades are directed and how they might interact with higher attentional processes. To pursue this goal, we are designing and building a closed-loop hardware system based on current models of the saccadic system.

Using traditional software methods to model neural systems is difficult because neural systems are composed of large numbers of elements with non-linear characteristics and a wide range of time-constants. Their mathematical behavior can rarely be solved analytically and simulations slow dramatically as the number and coupling of elements increases. Thus, real-time behavior, a critical issue for any system evolved for survival in a rapidly changing world, becomes impossible. Our approach to these problems has been to fabricate special purpose hardware that reflects the organization of real neural systems (Mead, 1989; Mahowald and Douglas, 1991; Horiuchi *et al.*, 1992.) Neuromorphic analog VLSI technology has many features in common with nervous tissue such as: processing strategies that are fast and reliable, circuits that are robust against noise and component variability, local parameter storage for the construction of adaptive systems and low-power consumption. Our analog chips and the nervous system both use low-accuracy components and are significantly constrained by wiring.

The design of the analog VLSI saccadic system discussed here is part of a long-term effort of a number of laboratories ( Douglas and Mahowald at Oxford University, Clark at Harvard University, Sejnowski at UCSD and the Salk Institute, Mead and Koch at Caltech) to design and build a complete replica of the early mammalian visual system in analog CMOS VLSI.

The design and fabrication of all circuits is carried out via the US-government sponsored silicon service MOSIS, using their 2 $\mu m$ line process.

## 2    An Analog VLSI Saccadic System

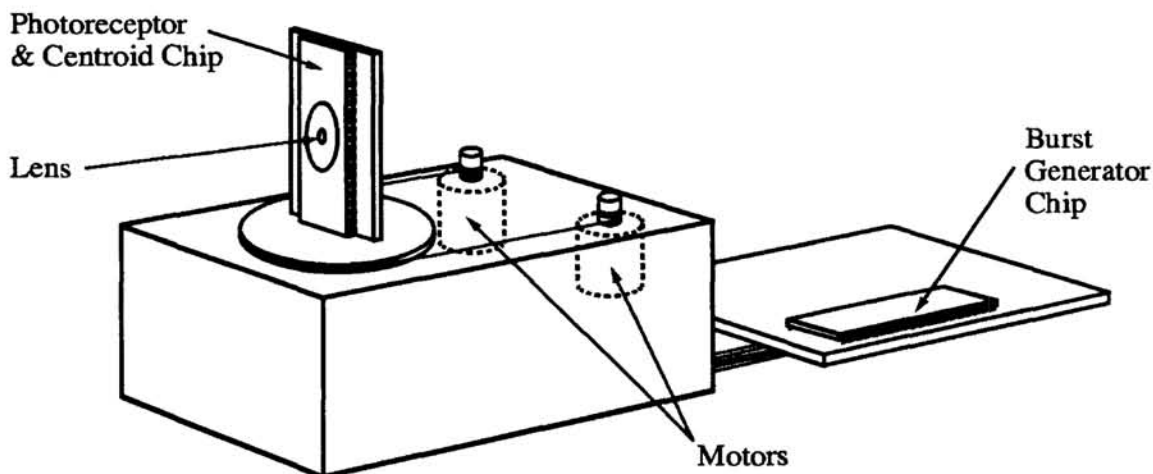

Figure 1: Diagram of the current system.

The system obtains visual inputs from a photoreceptor array, computes the target location within a model of the superior colliculus and outputs the saccadic burst command to drive the eyeball. While not discussed here, an auditory localization

input is being developed to trigger saccades to acoustical stimuli.

## 2.1   The Oculomotor Plant

The oculomotor plant is a one degree-of-freedom turntable which is driven by a pair of antagonistic-pulling motors. In the biological system where the agonist muscle pulls against a passive viscoelastic force, the fixation position is set by balancing these two forces. In our system, the viscoelastic properties of the oculomotor plant are simulated electronically and the fixation point is set by the shifting equilibrium point of these forces. In order maintain fixation off-center, like the biological system, a tonic signal to the motor controller must be maintained.

## 2.2   Photoreceptors

The front-end of the system is an adaptive photoreceptor array (Delbrück, 1992) which amplifies small changes in light intensity yet adapts quickly to gross changes in lighting level. The current system uses a 1-D array of 32 photoreceptors 40 microns apart. This array provides the visual input to the superior colliculus circuitry. The gain control occurs locally at each pixel of the image and thus the maximum sensitivity is maintained everywhere in the image in contrast to traditional imaging arrays which may provide washed out or blacked-out areas of an image when the contrast within an image is too large. In order to trigger reflexive, visually-guided saccades, the output of the photoreceptor array is coupled to the superior colliculus model by a luminance change detection circuit. A change in luminance somewhere in the image sends a pulse of current to the colliculus circuit which computes the center of this activity. This coupling passes a current signal which is proportional to the absolute-value of the temporal derivative of a photoreceptor's voltage output, (i.e. $\|dI(x,t)/dt\|$ where I(x,t) is the output of the photreceptor array). While we are initially building a 1-D system, 2-D photoreceptor arrays have been built in anticipation of a two degree-of-freedom system. While these photoreceptor circuits have been successfully constructed, we do not discuss the results here since the performance of these circuits are described in the literature (Delbrück 1992).

## 2.3   Superior Colliculus Model

The superior colliculus, located on the dorsal surface of the midbrain, is a key area in the behavioral orientation system of mammals. The superficial layers have a topographic map of visual space and the deeper layers contain a motor map of saccadic vectors. Microstimulation in this area initiates saccades whose metrics are related to the location stimulated. This type of representation is known as a **population coding**. Many neurons in the deeper layers of superior colliculus are multisensory and will generate saccades to auditory and somatosensory targets as well as visual targets.

While it is clear that the superior colliculus performs a multitude of integrative functions between sensory modalities and attentional processes, our initial model of superior colliculus simply computes the center of activity from the population code seen in the superficial layers (i.e. the photoreceptor array) using the weighted average techniques developed by DeWeerth (1991) for computing the centroid of

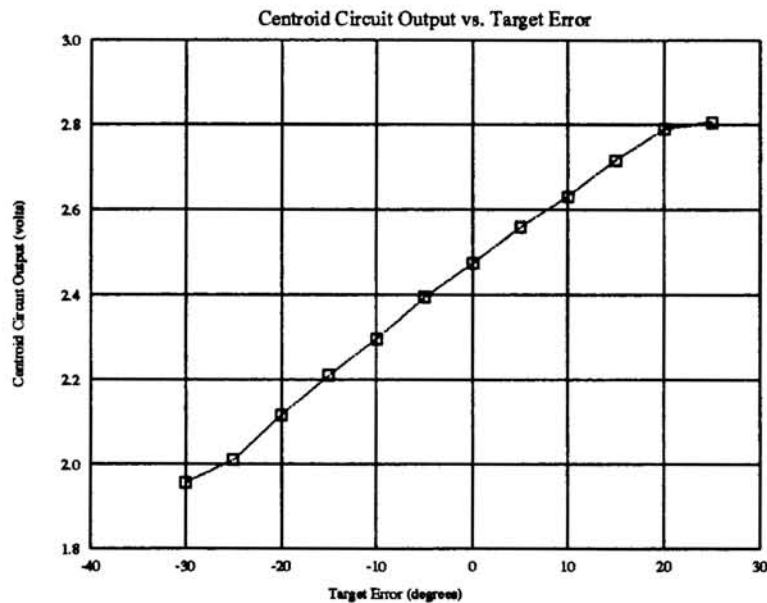

Figure 2: Output of the centroid circuit for a flashed red LED target at different angles away from the center position. Note that the output of the circuit was sampled 1 msec. after stimulus onset to account for capacitive delays.

brightness. The results of the photoreceptor/centroid circuits are shown in Fig. 2. In the case of visually-guided saccades, retinal error translates directly into motor error and thus we can use the photoreceptors directly as our inputs. This simplified retina/superior colliculus model provides the motor error which is then passed on to the burst generator.

## 2.4   Saccadic Burst Generator

The burst generator model (Fig. 3) driving the oculomotor plant receives as its input, desired change in eye position from the superior colliculus model and creates a two-component signal, a pulse and a step (Fig. 4). A pair of these pulse/step signals drive the two muscles of the eye which in turn moves the retinal array, thus closing the loop. The burst generator model is a double integrator model based on the work by Jürgens, *et al* (1981) and Lisberger *et al* (1987) which uses initial motor error as the input to the system. This motor error is injected into the "integrating" burst neuron which has negative feedback onto itself. This arrangement has the effect of firing a number of spikes proportional to the initial value of motor error. In the circuit, this integrator is implemented by a 1.9 pF capacitor. This burst of spikes serves to drive the eye rapidly against the viscosity. The burst is also integrated by the "neural integrator" (another 1.9 pF capacitor) which holds the local estimate of the current eye position from which the tonic, or holding signal is generated. Figs. 4 and 5 show output data from the burst generator chip and the response of the physical mechanism to this output. The inputs to the burst generator chip are 1) a voltage indicating desired eye position and 2) a digital trigger signal. The outputs are a pair of asynchronous digital pulse trains which carry the pulse/step signals which drive the left and right motors.

## 3    Discussion

As we are still in the formative stages of our project, our first goal has been to demonstrate a closed-loop system which can fixate a particular stimulus whose image is falling onto its photoreceptor array. The first set of chips represent dramatically simplified circuits in order to capture the first-order behavior of the system while using known representations. Owing to the large number of parameters that must be set, and their sensitivity to variations, we have begun a study to investigate biologically plausible approaches to automatic parameter-setting.

In the short term we intend to dramatically refine the models used at each stage, most notably the superior colliculus which is involved in the integration of non-visual sources of saccade targets (e.g. memory or audition), and in the mechanisms used for target selection or fixation. In the longer run, we plan to model the interaction of this system with other oculomotor processes such as smooth pursuit, VOR, OKR, AND vergence eye movements.

While the biological microcircuits of the superior colliculus and brainstem burst generator are not well known, more is understood about the representations found in these areas. By exploring the advantages and disadvantages of various computational models in a working system, it is hoped that a truly robust system will emerge as well as better models to explain the biological data. The construction of a compact hardware system which operates in real-time can often provide a more intuitive understanding of the closed-loop system. In addition, a visually-attentive hardware system which is physically small and low-power has numerous applications in the real world such as in mobile robotics or remote surveillance.

## 4    Acknowledgements

Many thanks go to Prof. Carver Mead and his group for developing the foundations of this research. Our laboratory is partially supported by grants from the Office of Naval Research and the Rockwell International Science Center. Tim Horiuchi is supported by a grant from the Office of Naval Research.

## 5    References

T. Delbrück and C. Mead, (1993) Ph.D. Thesis, California Institute of Technology.

S. P. DeWeerth, (1991) Ph.D. Thesis, California Institute of Technology.

T. Horiuchi, W. Bair, B. Bishofberger, A. Moore, J. Lazzaro, C. Koch, (1992) *Int. J. Computer Vision* **8**:3,203-216.

R. Jürgens, W. Becker, and H. H. Kornhuber, (1981) *Biol. Cybern.* **39**:87-96.

S. G. Lisberger, E. J. Morris, and L. Tychsen, (1987) *Ann. Rev. Neurosci.* **10**:97-129.

M. Mahowald, and R. Douglas, (1991) *Nature* **354**:515-518.

C. Mead, (1989) *Analog VLSI and Neural Systems*, Addison-Wesley.

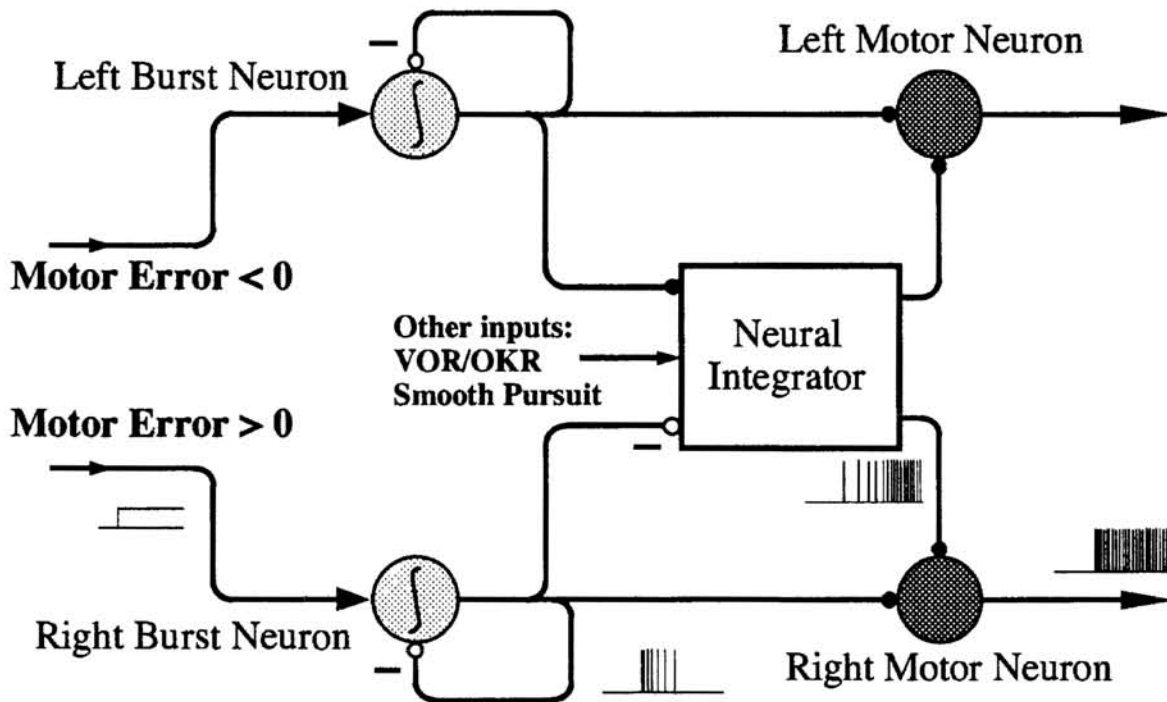

Figure 3: Schematic diagram of the burst generator. The burst neuron "samples" the motor error when it receives a trigger signal (not shown) and begins firing as a sigmoidal function of the motor error. The spikes feedback and discharge the integrator and the burst is shut down. This "pulse" signal drives the eye against the viscosity. This signal is also integrated by the neural integrator which contributes the "step" portion of the motor command to hold the eye in its final position. The neural integrator has additional velocity inputs for other oculomotor behavior such as smooth pursuit, VOR and OKR. Note that the burst neuron for the other muscle is silent in this direction.

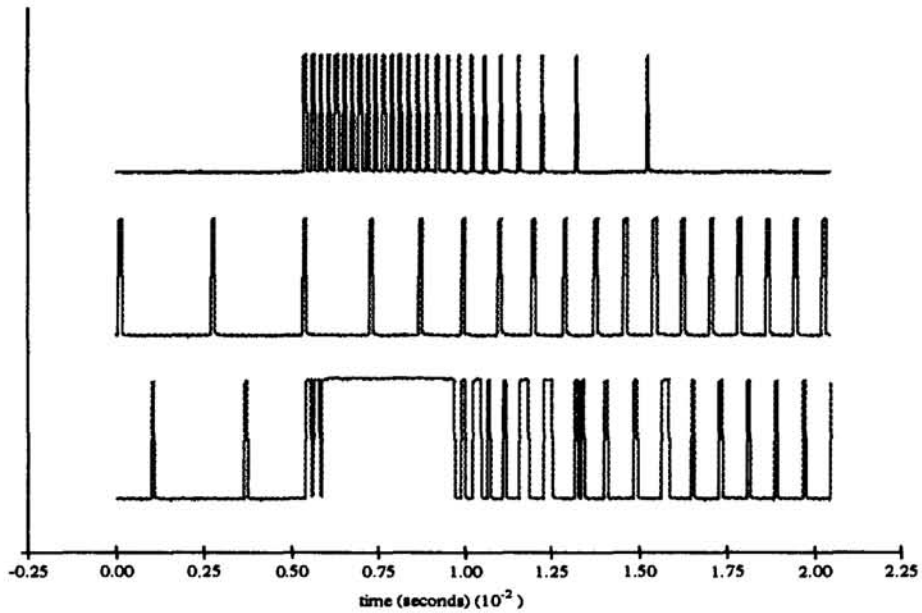

Figure 4: Spike signals in the circuit during a small saccade. (7.5 degrees to the right, starting from 4.8 degrees to the right.) Top: Burst neuron, Middle: Neural Integrator, Bottom: Motor neuron. (one of the outputs of the chip) Note that the "neuron" circuit currently used increases both its pulse frequency and pulse duration for large input currents, causing the voltage saturation seen in the bottom trace.

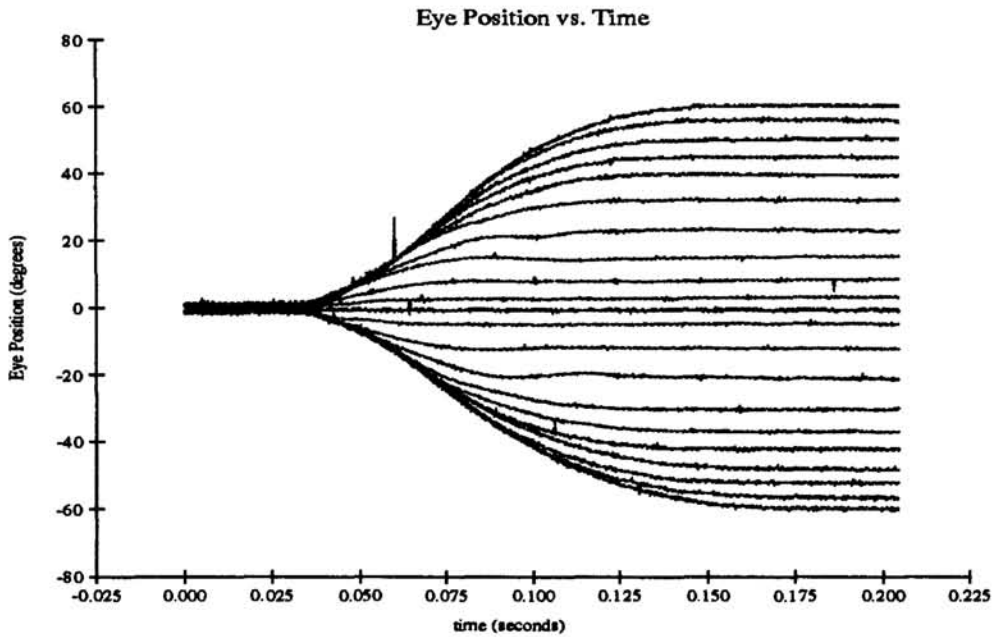

Figure 5: Horizontal position vs. time for 21 different saccades. Peak angular velocity achieved for the 60 degree saccade to the right was approximately 870 degrees per second. The input command was changed uniformly from -60 to +60 degrees.

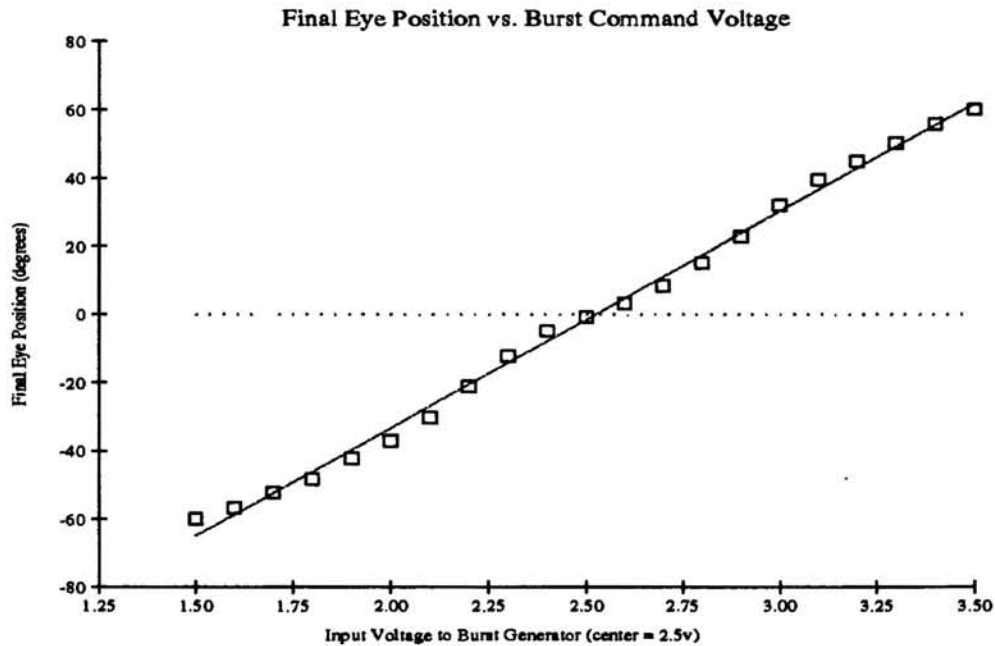

Figure 6: Linearity of the system for the position data given in the previous figure. Final eye position was computed as the average eye position during the last 20 msec. of each trace.

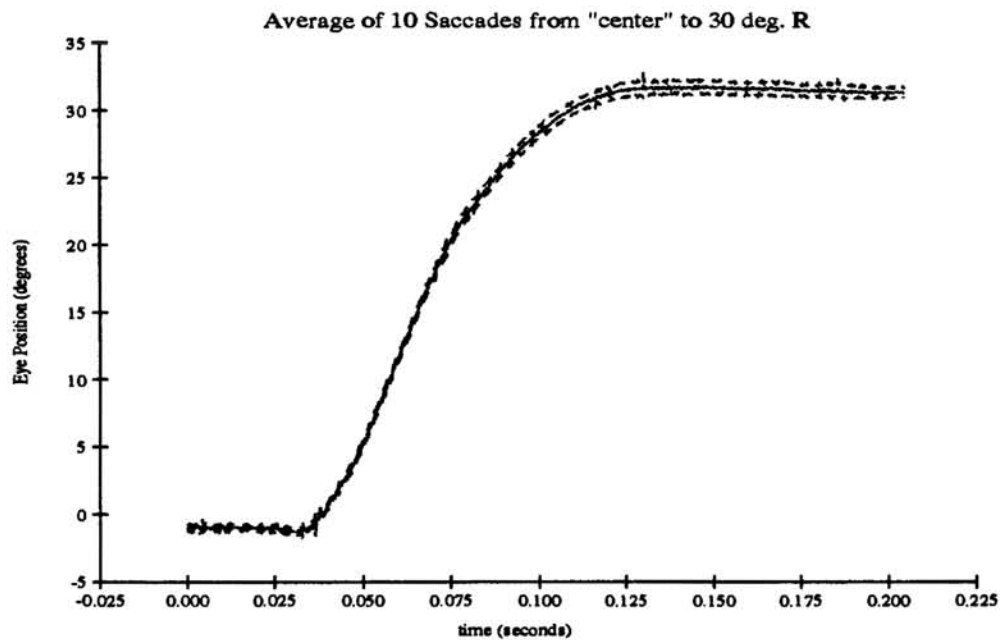

Figure 7: Repeatability: The solid line shows averaged eye position (relative to center) vs. time for 10 identical saccades. The dashed lines show a standard deviation on each side of the mean. Most of the variability is attributed to problems with friction.